# Maximum-Likelihood Continuity Mapping (MALCOM): An Alternative to HMMs

**David A. Nix**
dnix@lanl.gov
Computer Research & Applications
CIC-3, MS B265
Los Alamos National Laboratory
Los Alamos, NM 87545

**John E. Hogden**
hogden@lanl.gov
Computer Research & Applications
CIC-3, MS B265
Los Alamos National Laboratory
Los Alamos, NM 87545

## Abstract

We describe Maximum-Likelihood Continuity Mapping (MALCOM), an alternative to hidden Markov models (HMMs) for processing sequence data such as speech. While HMMs have a discrete "hidden" space constrained by a fixed finite-automaton architecture, MALCOM has a continuous hidden space—a *continuity map*—that is constrained only by a smoothness requirement on paths through the space. MALCOM fits into the same probabilistic framework for speech recognition as HMMs, but it represents a more realistic model of the speech production process. To evaluate the extent to which MALCOM captures speech production information, we generated continuous speech continuity maps for three speakers and used the paths through them to predict measured speech articulator data. The median correlation between the MALCOM paths *obtained from only the speech acoustics* and articulator measurements was 0.77 on an independent test set not used to train MALCOM or the predictor. This unsupervised model achieved correlations over speakers and articulators only 0.02 to 0.15 lower than those obtained using an analogous supervised method which *used articulatory measurements as well as acoustics.*.

## 1 INTRODUCTION

Hidden Markov models (HMMs) are generally considered to be the state of the art in speech recognition (e.g., Young, 1996). The strengths of the HMM framework include a rich mathematical foundation, powerful training and recognition algorithms for large speech corpora, and a probabilistic framework that can incorporate statistical phonology and syntax (Morgan & Bourlard, 1995). However, HMMs are known to be a poor model of the speech production process. While speech production is a continuous, temporally evolving process, HMMs treat speech production as a discrete, finite-state system where the current state depends only on the immediately preceding state. Furthermore, while HMMs are designed to capture temporal information as state transition probabilities, Bourlard *et al.*,

(1995) suggest that when the transition probabilities are replaced by constant values, recognition results do not significantly deteriorate. That is, while transitions are often considered the most perceptually relevent component of speech, the conventional HMM framework is poor at capturing transition information.

Given these deficiencies, we are considering alternatives to the HMM approach that maintain its strengths while improving upon its weaknesses. This paper describes one such model called Maximum-Likelihood Continuity Mapping (MALCOM). We first review a general statistical framework for speech recognition so that we can compare the HMM and MALCOM formulations. Then we consider what the abstract hidden state represents in MALCOM, demonstrating empirically that the paths through MALCOM's hidden space are closely related to the movements of the speech production articulators.

## 2   A GENERAL FRAMEWORK FOR SPEECH RECOGNITION

Consider an unknown speech waveform that is converted by a front-end signal-processing module into a sequence of acoustic vectors $\mathbf{X}$. Given a space of possible utterances, $W$, the task of speech recognition is to return the most likely utterance $W^*$ given the observed acoustic sequence $\mathbf{X}$. Using Bayes' rule this corresponds to

$$W^* = \underset{W}{\operatorname{argmax}} P(W|\mathbf{X}) = \underset{W}{\operatorname{argmax}} \frac{P(\mathbf{X}|W)P(W)}{P(\mathbf{X})}. \tag{1}$$

In recognition, $P(\mathbf{X})$ is typically ignored because it is constant over all $W$, and the posterior $P(W|\mathbf{X})$ is estimated as the product of the prior probability of the word sequence, $P(W)$, and the probability that the observed acoustics were generated by the word sequence, $P(\mathbf{X}|W)$. The prior $P(W)$ is estimated by a *language model*, while the production probability $P(\mathbf{X}|W)$ is estimated by an *acoustic model*. In continuous speech recognition, the product of these terms must be maximized over $W$; however, in this paper, we will restrict our attention to the form of the acoustical model only. Every candidate utterance $W$ corresponds to a sequence of word/phone models $\mathcal{M}_w$ such that $P(\mathbf{X}|W) = P(\mathbf{X}|\mathcal{M}_w)$, and each $\mathcal{M}_w$ considers all possible paths through some "hidden" space. Thus, for each candidate utterance, we must calculate

$$P(\mathbf{X}|\mathcal{M}_w) = \int_{\mathbf{Y}} P(\mathbf{X}|\mathbf{Y}, \mathcal{M}_w)P(\mathbf{Y}|\mathcal{M}_w)d\mathbf{Y}, \tag{2}$$

where $\mathbf{Y}$ is some path through the hidden space.

### 2.1   HIDDEN MARKOV MODELS

Because HMMs are finite-state machines with a given fixed architecture, the path $\mathbf{Y}$ through the hidden space corresponds to series of discrete states, simplifying the integral of Eq. (2) to a sum. However, to avoid computing the contribution of all possible paths, the *Viterbi approximation*—considering only the single path that maximizes Eq. (2)—is frequently used without much loss in recognition performance (Morgan & Bourlard, 1995). Thus,

$$P(\mathbf{X}|\mathcal{M}_w) \approx \underset{\mathbf{Y}}{\operatorname{argmax}} P(\mathbf{X}|\mathbf{Y}, \mathcal{M}_w)P(\mathbf{Y}|\mathcal{M}_w). \tag{3}$$

The first term corresponds to the product of the emission probabilities of the acoustics given the state sequence and is typically estimated by mixtures of high-dimensional Gaussian densities. The second term corresponds to the product of the state transition probabilities. However, because Bourlard *et al.* (1995) found that this second term contributes little to recognition performance, the modeling power of the conventional HMM must reside in the first term. Training the HMM system involves estimating both the emission and the

transition probabilities from real speech data. The Baum-Welch/forward-backward algorithm (e.g., Morgan & Scofield, 1994) is the standard computationally efficient algorithm for iteratively estimating these distributions.

## 2.2 MAXIMUM-LIKELIHOOD CONTINUITY MAPPING (MALCOM)

In contrast to HMMs, the multi-dimensional MALCOM hidden space is continuous—there are an infinite number states and paths through them. While the HMM is constrained by a fixed architecture, MALCOM is constrained by the notion of continuity of the hidden path. That is, the path must be smooth and continuous: it may not carry any energy above a given cutoff frequency. Unlike the discrete path in an HMM, the smooth hidden path in MALCOM attempts to emulate the motion of the speech articulators in what we call a *continuity map* (CM).

Unless we know how to evaluate the integral of Eq. (2) (which we currently do not), we must also make the Viterbi approximation and approximate $P(\mathbf{X}|\mathcal{M}_w)$ by considering only the single path that maximizes the likelihood of the acoustics $\mathbf{X}$ given the utterance model $\mathcal{M}_w$, resulting in Eq. (3) once again. Analogously, the first term, $P(\mathbf{X}|\mathbf{Y}, \mathcal{M}_w)$, corresponds to the acoustic generation probability given the hidden path, and the second term corresponds to the probability of the hidden path given the utterance model. This paper focuses on the first term because this is the term that produces conventional HMM performance.[1]

Common to all $\mathcal{M}_w$ is a set of $N$ probability density functions (pdfs) $\Phi$ that define the CM hidden space, modeling the likelihood of $\mathbf{Y}$ given $\mathbf{X}$ for an $N$-code vector quantization (VQ) of the acoustic space. Because these pdfs are defined over the low-dimensional CM space instead of the high-dimensional acoustic space (e.g., 6 vs. 40+), MALCOM requires many fewer parameters to be estimated than the corresponding HMM.

## 3  THE MALCOM ALGORITHM

We now turn to developing an algorithm to estimate both the CM pdfs $\Phi$ and the corresponding paths $\mathbf{Y}$ that together maximize the likelihood of a given time series of acoustics, $\mathcal{L} = P(\mathbf{X}|\mathbf{Y}, \Phi)$. This is an extension of the method first proposed by Hogden (1995), in which he instead maximized $P(\mathbf{Y}|\mathbf{X}, \Phi)$ using vowel data from a single speaker. Starting with random but smooth $\mathbf{Y}$, the MALCOM training algorithm generates a CM by iterating between the following two steps: (1) Given $\mathbf{Y}$, reestimate $\Phi$ to maximize $\mathcal{L}$; and (2) Given $\Phi$, reestimate smooth paths $\mathbf{Y}$ to maximize $\mathcal{L}$.

### 3.1  LOG LIKELIHOOD FUNCTION

To specify the log likelihood function $\mathcal{L}$, we make two dependence claims and one independence assumption. First we claim that $\mathbf{y}_t$ depends (to at least some small extent) on all other $\mathbf{y}$ in the utterance, an expression of the continuity constraint described above. We make another natural claim that $\mathbf{x}_t$ depends on $\mathbf{y}_t$, that the path configuration at time $t$ influences the corresponding acoustics. However, we do make the conditional independence assumption that

$$\mathcal{L} = P(\mathbf{X}|\mathbf{Y}, \Phi) = \prod_{t=1}^{n} P(\mathbf{x}_t|\mathbf{y}_t, \Phi). \tag{4}$$

Note that Eq. (4) does not assume that each $\mathbf{x}_t$ is independent of $\mathbf{x}_{t-1}$ (as is often assumed in data modeling); it only assumes that the *conditioning* of $\mathbf{x}_t$ on $\mathbf{y}_t$ is independent from

$t-1$ to $t$. For example, because $\mathbf{x}_t$ depends on $\mathbf{y}_t$, $\mathbf{y}_t$ depends on all other $\mathbf{y}$ (the smoothness constraint), and $\mathbf{x}_{t-1}$ depends on $\mathbf{y}_{t-1}$, $\mathbf{x}_t$ is *not* assumed to be independent of all other $\mathbf{x}$s in the utterance.

With a log transformation and an invocation of Bayes' rule, we obtain the MALCOM log likelihood function:

$$\ln \mathcal{L} = \sum_{t=1}^{n} \left[ \ln P(\mathbf{y}_t|\mathbf{x}_t, \Phi) + \ln P(\mathbf{x}_t) - \ln P(\mathbf{y}_t|\Phi) \right]. \tag{5}$$

We model each $P(\mathbf{y}_t|\mathbf{x}_t, \Phi)$ by a probability density function (pdf) $p[\mathbf{y}_t|\mathbf{x}_t, \Phi_j(\mathbf{x}_t)]$, where the particular model $\Phi_j$ depends on which of the $N$ VQ codes $\mathbf{x}_t$ is assigned to. Here we use a simple multi-dimensional Gaussian for each pdf, but we are currently exploring the use of multi-modal mixtures of Gaussians to represent the pdfs for sounds such as stop consonants for which the inverse map from acoustics to articulation may not be unique (Nix, 1998). Next, we need an estimate of $P(\mathbf{y}_t|\Phi)$, which can be obtained by summing over all VQ partitions: $P(\mathbf{y}_t|\Phi) \approx \sum_{j=1}^{N} p(\mathbf{y}_t|\mathbf{x}_j, \Phi_j)P(\mathbf{x}_j)$. We estimate $P(\mathbf{x}_j)$ by calculating the relative frequency of each acoustic code in the VQ codebook.

## 3.2 PDF ESTIMATION

For step (1) of training, we use gradient-based optimization to reestimate the means of the Gaussian pdfs for each acoustic partition, where the gradient of Eq.(5) with respect to the mean of pdf $i$ is

$$\nabla_{\boldsymbol{\mu}_i} \ln \mathcal{L} = \sum_{t \in \mathbf{x}(t)=\mathbf{x}_i} \Sigma_i^{-1}(\mathbf{y}_t - \boldsymbol{\mu}_i) - \sum_{i=1}^{t} \left\{ \frac{\sum_{j=1}^{N} p[\mathbf{y}_t|\mathbf{x}_j, \Phi(\mathbf{x}_j)]P(\mathbf{x}_j)\Sigma_j^{-1}(\mathbf{y}_t - \boldsymbol{\mu}_j)}{\sum_{j=1}^{N} p[\mathbf{y}_t|\mathbf{x}_j, \Phi(\mathbf{x}_j)]P(\mathbf{x}_j)} \right\}$$
$$\tag{6}$$

where $\Sigma$ is the covariance matrix for each pdf. For the results in this paper, we use a common radially symmetric covariance matrix for all pdfs and reestimate the covariance matrix after each path optimization step.[2] In doing the optimization, we employ the following algorithm:

1. Make an initial guess of each $\boldsymbol{\mu}_i$ as the means of the path configurations corresponding to the observed acoustics $\mathbf{X} \in \mathbf{x}_i$.
2. Construct $\nabla_{\boldsymbol{\mu}} \ln \mathcal{L}$ by considering Eq. (6) over all $N$ acoustic partitions.
3. Determine a search direction for the optimization using, for example, conjugate gradients and perform a line search along this direction (Press *et al.*, 1988).
4. Repeat steps [2]–[3] until convergence.

To avoid potential degenerate solutions, after each pdf optimization step, the dimensions of the CM are orthogonalized. Furthermore, because the scale of the continuity map is meaningless (only its topological arrangement matters), the $N$ pdf means are scaled to zero mean, unit variance before each path optimization step.

## 3.3 PATH ESTIMATION

For step (2) of training, we use gradient-based optimization to reestimate $\mathbf{Y}$, where the gradient of the log likelihood function with respect to a specific $\mathbf{y}_t$ is given by

$$\nabla_{\mathbf{y}_t} \ln \mathcal{L} = \frac{\nabla_{\mathbf{y}_t} p[\mathbf{y}_t|\mathbf{x}_t, \Phi(\mathbf{x}_t)]}{p[\mathbf{y}_t|\mathbf{x}_t, \Phi(\mathbf{x}_t)]} - \frac{\nabla_{\mathbf{y}_t} \sum_{j=1}^{N} p[\mathbf{y}_t|\mathbf{x}_j, \Phi(\mathbf{x}_j)]P(\mathbf{x}_j)}{\sum_{j=1}^{N} p[\mathbf{y}_t|\mathbf{x}_j, \Phi(\mathbf{x}_j)]P(\mathbf{x}_j)}. \tag{7}$$

In doing the optimization, we employ the following gradient-based algorithm:

1. Make an initial guess of the path $\mathbf{Y}^0$ as the means of the pdfs corresponding to the observed acoustic sequence $\mathbf{X}$.
2. Low pass filter $\mathbf{Y}^0$.
3. Construct $\nabla_\mathbf{Y} \ln \mathcal{L}$ by considering Eq. (7) over all $t$.
4. Determine a search direction for the optimization using, for example, conjugate gradients (Press *et al.*, 1988).
5. Low-pass filter this search direction using the same filter as in step [2].
6. Perform a line search along the filtered direction (Press *et al.*, 1988).
7. Repeat steps [3]–[6] until convergence.

Because neither the line search direction nor the initial estimate $\mathbf{Y}^0$ contains energy above the cutoff frequency of the low-pass filter, their linear addition—the next estimate of $\mathbf{Y}$—will not contain energy above the cutoff frequency either. Thus, steps [2] and [5] implement the desired smoothness constraint.

## 4  COMPARNG MALCOM PATHS TO SPEECH ARTICULATION

To evaluate our claim that MALCOM paths are topologically related to articulator motions, we construct a regression predictor from $\mathbf{Y}$ to measured articulator data using the training data and test the quality of this predictor on an independent test set.

Our speech corpus consists of data from two male and one female native speakers of German. This data was obtained from Dr. Igor Zlokarnik and recorded at the Technical University of Munich, Germany using electro-magnetic articulography (EMA) (Perkell *et al.*, 1992). Each speaker's articulatory measurements and acoustics were recorded for the same 108 sentences, where each sentence was about 4 seconds long.

The acoustics were recorded using a room-placed microphone and sampled using 16-bit resolution at 16 kHz. Prior to receiving the data from Munich, the data were resampled at 11025 Hz. To represent the acoustic signal in compact vector time-series, we used 256-sample (23.2 msec) Hamming-windowed frames, with a new frame starting every 5.8 msec (75% overlap). We transform each frame into a 13th-order LPC-cepstral coefficient vector $\mathbf{a}_t$ (12 cepstral features plus log gain—see Morgan& Scofield, 1994). A full acoustical feature vector $\mathbf{x}_t$ consists of a window of seven frames such that $\mathbf{x}_t$ is made up of the frames $\{\mathbf{a}_{t-6}, \mathbf{a}_{t-4}, \mathbf{a}_{t-2}, \mathbf{a}_t, \mathbf{a}_{t+2}, \mathbf{a}_{t+4}, \mathbf{a}_{t+6}\}$. To VQ the acoustic space we used the classical $k$-means algorithm (e.g., Bishop, 1995), but we used 512 codes to model the vowel data, and 256 codes each to model the stop consonants, the fricatives, the nasals, and the liquids (1536 codes combined).[3]

The articulatory data consist of the $(x, y)$ coordinates of 4 coils along the tongue and the $y$-coordinates of coils on the jaw and lower lip. Figure 1 illustrates the approximate location of each coil. The data were originally sampled at 250 Hz but were resampled to 172.26 Hz to match one articulatory sample for each 75%-overlapping acoustic frame of 256 samples. The articulatory data were subsequently low-pass filtered at 15 Hz to remove measurement noise.

Sentences 1–90 were used as a training set, and sentences 91–108 were withheld for evaluation. A separate CM was generated for each speaker using the training data. We used an 8 Hz cutoff frequency because the measured articulatory data had very little energy above 8 Hz, and a 6-dimensional continuity map was used because the first six principal components capture 99% of the variance of the corresponding articulator data (Nix, 1998).

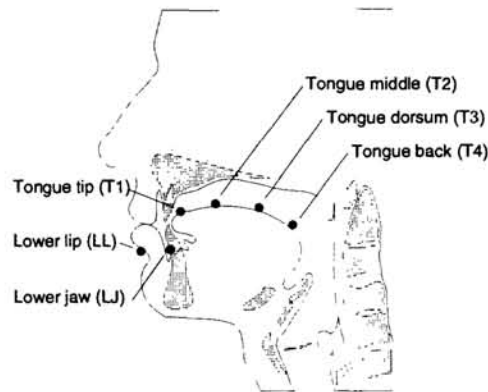

Figure 1: Approximate positions of EMA coils for speech articulation measurements.

Because the third term in Eq. (5) is computationally complex, we approximated Eq. (5) by only its first term (the second term is constant during training) until $\ln \mathcal{L}$, calculated at the end of each iteration using all terms, started to decrease. At this point we started using both the first and third terms of Eq. (5). In each pdf and path optimization step, our convergence criterion was when the maximum movement of a mean or a path was $< 10^{-4}$. Our convergence criterion for the entire algorithm was when the correlation of the paths from one full iteration of pdf and path optimization to another was $> 0.99$ in all dimensions. This usually took about 30 iterations.

To evaluate the extent to which MALCOM hidden paths capture information related to articulation, we used the same training set to estimate a non-linear regression function from the output generated by MALCOM to the corresponding measured articulator data. We used an ensemble of 10 single-hidden-layer, 32-hidden unit, multi-layer perceptrons trained on different 2/3-training, 1/3-early stopping partitions of the training set, where the results of the ensemble on the test set were averaged (e.g., Bishop, 1995). A linear regression produced results approximately 10% worse than those we report here.

To contrast with the unsupervised MALCOM method, we also tested a supervised method in which the articulatory data was available for training as well as evaluation. This involved only the pdf optimization step of MALCOM because the paths were fixed as the articulator measurements. The resulting pdfs were then used in the path optimization step to determine paths for the test data acoustics. We could then measure what fraction of this supervised performance the unsupervised MALCOM attained.

## 5   RESULTS AND CONCLUSIONS

The results of this regression on the test set are plotted in Figure 2. The MALCOM paths had a median correlation of 0.77 with the actual articulator data, compared to 0.84 for the comparable supervised method. Thus, *using only the speech acoustics*, MALCOM generated continuity maps with correlations to real articulator measurements only 0.02 to 0.15 lower than the corresponding supervised model which *used articulatory measurements as well as acoustics*.

Given that (1) MALCOM fits into the same probabilistic framework for speech recognition as HMMs and (2) MALCOM's hidden paths capture considerable information about the speech production process, we believe that MALCOM will prove to be a viable alternative to the HMM for speech processing tasks. Our current work emphasizes developing a word model to complete the MALCOM formulation and test a full speech recognition system. Furthermore, MALCOM is applicable to any other task to which HMMs can be applied,

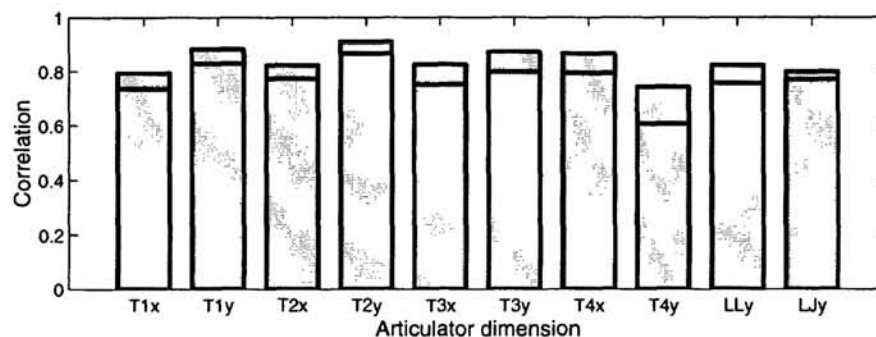

Figure 2: Correlation between estimated and actual articulator trajectories on the independent test set averaged across speakers. Each full bar is the performance of the supervised analogy to MALCOM, and the horizontal line on each bar is the performance of MALCOM itself.

including fraud detection (Hogden, 1997) and text processing.

## Acknowledgments

We would like to thank James Howse and Mike Mozer for their helpful comments on this manuscript and Igor Zlokarnik for sharing his data with us. This work was performed under the auspices of the U.S. Department of Energy.

## Footnotes

[1]However, we are currently developing a model of $P(\mathbf{Y}|\mathcal{M}_w)$ to replace the corresponding (and useless) term in the conventional HMM formulation as well (Hogden *et al.*, 1998).

[2]However, we are currently exploring the effects of individual and diagonal covariance matrices.

[3]This acoustic representation and VQ scheme were determined to work well for modeling real articulator data (Nix, 1998), so they were used here as well.

## References

Bishop, C.M. (1995). *Neural Networks for Pattern Recognition*, NY: Oxford University Press, Inc.

Bourlard, H. Konig, Y., & Morgan, N. (1995). "REMAP: Recursive estimation and maximization of a posteriori probabilities, application to transition-based connectionist speech recognition," International Computer Science Institute Technical Report TR-94-064.

Hogden, J. (1995). "Improving on hidden Markov models: an articulatorily constrained, maximum-likelihood approach to speech recognition and speech coding," Los Alamos National Laboratory Technical Report, LA-UR-96-3945.

Hogden, J. (1997). "Maximum likelihood continuity mapping for fraud detection," Los Alamos National Laboratory Technical Report, LA-UR-97-992.

Hogden, J., Nix, D.A., Gracco, V., & Rubin, P. (1998). "Stochastic word nodels for articulatorily constrained speech recognition and synthesis," submitted to Acoustical Society of America Conference, 1998.

Morgan, N. & Bourlard, H.A. (1995). "Neural Networks for Statistical Recognition of Continuous Speech," *Proceedings of the IEEE*, **83**(5), 742–770.

Morgan, D.P., & Scofield, C.L. (1992). *Neural Networks and Speech Processing*, Boston, MA: Kluwer Academic Publishers.

Nix, D.A. (1998). *Probabilistic methods for inferring vocal-tract articulation from speech acoustics*, Ph.D. Dissertation, U. of CO at Boulder, Dept. of Computer Science, in preparation.

Perkell, J.S., Cohen, M.H., Svirsky, M.A., Matthies, M.L., Garabieta, I., & Jackson, M.T.T. (1992). "Electromagnetic midsagittal articulometer systems for transducing speech articulatory movements," *Journal of the Acoustical Society of America*, **92**(6), 3078–3096.

Press, W.H., Teukolsky, S.A., Vetterling, W.T., & Flannery, B.P. (1988). *Numerical Recipes in C* Cambridge University Press.

Young, S.J. (1996). "A review of large-vocabulary continuous speech recognition," *IEEE Signal Processing Magazine*, September, 45–57.